# Boosting versus Covering

**Kohei Hatano**[*]
Tokyo Institute of Technology
hatano@is.titech.ac.jp

**Manfred K. Warmuth**
UC Santa Cruz
manfred@cse.ucsc.edu

## Abstract

We investigate improvements of AdaBoost that can exploit the fact that the weak hypotheses are one-sided, i.e. either all its positive (or negative) predictions are correct. In particular, for any set of $m$ labeled examples consistent with a disjunction of $k$ literals (which are one-sided in this case), AdaBoost constructs a consistent hypothesis by using $O(k^2 \log m)$ iterations. On the other hand, a greedy set covering algorithm finds a consistent hypothesis of size $O(k \log m)$. Our primary question is whether there is a simple boosting algorithm that performs as well as the greedy set covering.

We first show that InfoBoost, a modification of AdaBoost proposed by Aslam for a different purpose, does perform as well as the greedy set covering algorithm. We then show that AdaBoost requires $\Omega(k^2 \log m)$ iterations for learning $k$-literal disjunctions. We achieve this with an adversary construction and as well as in simple experiments based on artificial data. Further we give a variant called SemiBoost that can handle the degenerate case when the given examples all have the same label. We conclude by showing that SemiBoost can be used to produce small conjunctions as well.

## 1 Introduction

The boosting method has become a powerful paradigm of machine learning. In this method a highly accurate hypothesis is built by combining many "weak" hypotheses. AdaBoost [FS97, SS99] is the most common boosting algorithm. The protocol is as follows. We start with $m$ labeled examples labeled with $\pm 1$. AdaBoost maintains a distribution over the examples. At each iteration $t$, the algorithm receives a $\pm 1$ valued weak hypothesis $h_t$ whose error (weighted by the current distribution on the examples) is slightly smaller than $\frac{1}{2}$. It then updates its distribution so that after the update, the hypothesis $h_t$ has weighted error exactly $\frac{1}{2}$. The final hypothesis is a linear combination of the received weak hypotheses and it stops when this final hypothesis is consistent with all examples.

It is well known [SS99] that if each weak hypothesis has weighted error at most $\frac{1}{2} - \frac{\gamma}{2}$, then the upper bound on the training error reduces by a factor of $\sqrt{1 - \gamma^2}$

---

[*]This research was done while K. Hatano was visiting UC Santa Cruz under the EAP exchange program.

and after $O(\frac{1}{\gamma^2} \log m)$ iterations, the final hypothesis is consistent with all examples. Also, it has been shown that if the final hypotheses are restricted to (unweighted) majority votes of weak hypotheses [Fre95], then this upper bound on the number of iterations cannot be improved by more than a constant factor.

However, if there always is a *positively one-sided* weak hypothesis (i.e. its positive predictions are always correct) that has error[1] at most $\frac{1}{2} - \frac{\gamma}{2}$, then a set cover algorithm can be used to reduce the training error by a factor[2] of $1 - \gamma$ and $O(\frac{1}{\gamma} \log m)$ weak hypotheses suffice to form a consistent hypothesis [Nat91]. In this paper we show that the improved factor is also achieved by InfoBoost, a modification of AdaBoost developed by Aslam [Asl00] based on a different motivation.

In particular, consider the problem of finding a consistent hypothesis for $m$ examples labeled by a $k$ literal disjunction. Assume we use the literals as the pool of weak hypotheses and always choose as the weak hypothesis a literal that is consistent with all negative examples. Then it can be shown that, for any distribution $D$ on the examples, there exists a literal (or a constant hypothesis) $h$ with weighted error at most $\frac{1}{2} - \frac{1}{4k}$ (See e.g. [MG92]). Therefore, the upper bound on the training error of AdaBoost reduces by a factor of $\sqrt{1 - \frac{1}{4k^2}}$ and $O(k^2 \log m)$ iterations suffice.

However, a trivial greedy set covering algorithm, that follows a strikingly similar protocol as the boosting algorithms, finds a consistent disjunction with $O(k \log m)$ literals. We show that InfoBoost mimics the set cover algorithm in this case (and attains the improved factor of $1 - \frac{1}{k}$).

We first explain the InfoBoost algorithm in terms of constraints on the updated distribution. We then show that $\Omega(k^2 \log m)$ iterations are really required by AdaBoost using both an explicit construction (which requires some assumptions) and artificial experiments. The differences are quite large: For $m = 10,000$ random examples and a disjunction of size $k = 60$, AdaBoost requires 2400 iterations (on the average), whereas Covering and InfoBoost require 60 iterations. We then show that InfoBoost has the improved reduction factor if the weak hypotheses happen to be one-sided. Finally we give a modified version of AdaBoost that exploits the one-sidedness of the weak hypotheses and avoids some technical problems that can occur with InfoBoost. We also discuss how this algorithm can be used to construct small conjunctions.

## 2 Minimizing relative entropy subject to constraints

Assume we are given a set of $m$ examples $(x_1, y_1), \ldots, (x_m, y_m)$. The instances $x_i$ are in some domain $\mathcal{X}$ and the labels $y_i$ are in $\{-1, 1\}$. The boosting algorithms maintain a distribution $D_t$ over the examples. The initial distribution is $D_1$ and is typically uniform. At the $t$-th iteration, the algorithm chooses a weak[3] hypothesis $h_t : \mathcal{X} \to \{-1, 1\}$ and then updates its distribution. The most popular boosting algorithm does this as follows:

$$\textbf{AdaBoost: } D_{t+1}(i) = \frac{D_t(i) \exp\{-y_i h_t(x_i)\alpha_t\}}{Z_t},$$

Here $Z_t$ is a normalization constant and the coefficient $\alpha_t$ depends on the *error* $\epsilon_t$ at iteration $t$: $\alpha_t = \frac{1}{2} \ln \frac{1-\epsilon_t}{\epsilon_t}$ and $\epsilon_t = \Pr_{D_t}[h_t(x_i) \neq y_i]$. The final hypothesis is given by the sign of the following linear combination of the chosen weak hypotheses: $H(x) = \sum_{t=1}^{T} \alpha_t h_t(x)$. Following [KW99, Laf99], we motivate the updates on the distributions of boosting algorithms as a constraint minimization of the relative entropy between the new and old distributions:

$$\textbf{AdaBoost: } D_{t+1} = \operatorname{argmin}_{D \in [0,1]^m, \sum_i D(i)=1} \Delta(D, D_t), \text{ s.t. } \Pr_D[h_t(x_i) \neq y_i] = \frac{1}{2}.$$

Here the relative entropy is defined as $\Delta(D, D') = \sum_i D(i) \ln \frac{D(i)}{D'(i)}$ and error w.r.t. the updated distribution is constraint to half.

The constraint can be easily understood using the table of Figure 1. There are two types of misclassified examples: false positive (weight c) and false negative (weight b). The AdaBoost constraint means $b + c = \frac{1}{2}$ w.r.t. the updated distribution $D_{t+1}$.

| $y_i \setminus h_t$ | +1 | −1 |
|---|---|---|
| +1 | $a$ | $b$ |
| −1 | $c$ | $d$ |

**Figure 1:** Four types of examples.

The second boosting algorithm we discuss in this paper has the following update:

$$\textbf{InfoBoost: } D_{t+1}(i) = \frac{D_t(i) \exp\{-y_i h_t(x_i) \alpha_t[h_t(x_i)]\}}{Z_t},$$

where $\alpha_t[\pm 1] = \frac{1}{2} \ln \frac{1-\epsilon[\pm 1]}{\epsilon_t[\pm 1]}$, $\epsilon_t[\pm 1] = \Pr_{D_t}[h_t(x_i) \neq y_i | h_t(x_i) = \pm 1]$ and $Z_t$ is the normalization factor. The final hypothesis is given by the sign of $H(x) = \sum_{t=1}^{T} \alpha_t[h_t(x)] \, h_t(x)$.

In the original paper [Asl00], the InfoBoost update was motivated by seeking a distribution $D_{t+1}$ for which the error of $h_t$ is half and $y_i$ and $h_t(x_i)$ have mutual information zero. Here we motivate InfoBoost as a minimization of the same relative entropy subject to the AdaBoost constraint $b + c = \frac{1}{2}$ and a second simultaneously enforced constraint $a + b = \frac{1}{2}$. Note that the second constraint is the AdaBoost constraint w.r.t. the constant hypothesis **1**. A natural question is why not just do two steps of AdaBoost at each iteration $t$: One for $h_t$ and and then, sequentially, one for **1**. We call the latter algorithm *AdaBoost with Bias*, since the constant hypothesis introduces a bias into the final hypothesis. See Figure 2 for an example of the different updates.

$D_t$ :

| $y_i \setminus h_t$ | +1 | −1 |
|---|---|---|
| +1 | $\frac{2}{5}$ | $\frac{2}{5}$ |
| −1 | $0$ | $\frac{1}{5}$ |

$D_{t+1}$ : **AdaB.**

| $y_i \setminus h_t$ | +1 | −1 |
|---|---|---|
| +1 | $\frac{1}{3}$ | $\frac{1}{2}$ |
| −1 | $0$ | $\frac{1}{6}$ |

$D_{t+1}$ : **InfoB.**

| $y_i \setminus h_t$ | +1 | −1 |
|---|---|---|
| +1 | $0$ | $\frac{1}{2}$ |
| −1 | $0$ | $\frac{1}{2}$ |

$D_{t+1}$ : **AdaB.w.Bias**

| $y_i \setminus h_t$ | +1 | −1 |
|---|---|---|
| +1 | $\frac{1}{5}$ | $\frac{3}{10}$ |
| −1 | $0$ | $\frac{1}{2}$ |

Figure 2: Updating based on a positively one-sided hypothesis $h_t$ (weight c is 0): The updated distributions on the four types of examples are quite different.

We will show in the next section that in the case of learning disjunctions, AdaBoost with Bias (and plain AdaBoost) can require many more iterations than InfoBoost and the trivial covering algorithm. This is surprising because the AdaBoost with Bias and InfoBoost seem so similar to each other (simultaneous versus sequential

enforcement of the same constraints). A natural extension would be to constrain the errors of all past hypotheses to half which is the *Totally Corrective Algorithm* of [KW99]. However this can lead to subtle convergence problems (See discussion in [RW02]).

## 3   Lower bounds of AdaBoost for Learning $k$ disjunctions

So far we did not specify how the weak hypothesis $h_t$ is chosen at iteration $t$. We assume there is a pool $\mathcal{H}$ of weak hypotheses and distinguish two methods:
**Greedy:** Choose a $h_t \in \mathcal{H}$ for which the normalization factor $Z_t$ in the update of the algorithm is minimized.
**Minimal:** Choose $h_t$ with error smaller than a given threshold $\frac{1}{2} - \delta$.
The greedy method is motivated by the fact that $\prod_t Z_t$ upper bounds the training error of the final hypothesis ([SS99, Asl00]) and this method greedily minimizes this upper bound. Note that the $Z_t$ factors are different for AdaBoost and InfoBoost.

In our lower bounds on the number of iterations the example set is always consistent with a $k$-literal monotone disjunction over $N$ variables. More precisely the instances $\boldsymbol{x}_i$ are in $\{\pm 1\}^N$ and the label $y_i$ is $x_{i,1} \vee x_{i,2} \vee \ldots \vee x_{i,k}$. The pool of weak learners consists of the $N$ literals $X_j$, where $X_j(\boldsymbol{x}_i) = x_{ij}$. For the greedy method we show that on random data sets InfoBoost and the covering algorithm use drasti-

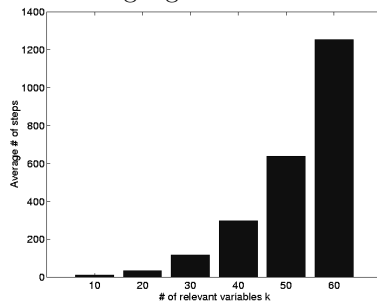

cally fewer iterations than AdaBoost with Bias. We chose $10,000$ examples as follows: The first $k$ bits of each example are chosen independently at random so that the probability of label $+1$ is half (i.e. the probability of $+1$ for each of the first $k$ bits is $1 - 2^{-1/k}$); the remaining $N - k$ irrelevant bits of each example are chosen $+1$ with probability half. Figure 3 shows the number of iterations as function of the size of disjunction $k$ (averaged over 20 runs) of AdaBoost with Bias until consistency is reached on all $10,000$ exam- **Figure 3:** Average # of steps ples. The number of iteration in this very simple of AdaBoost with Bias for $k =$ setting grows quadratically with $k$. If the num- $10, 20, 30, 40, 50, 60$. ber of iterations is divided by $k^2$ then the resulting curve is larger than a constant. In contrast the number of iterations of the greedy covering algorithm and InfoBoost is provably linear in $k$: For $k = 60$ and $m = 10,000$, the former require 60 iterations on the average, whereas AdaBoost with Bias with the greedy choice of the weak hypothesis requires 1200 even though it never chooses irrelevant variables as weak learners (Plain AdaBoost requires twice as many iterations).

The above construction is not theoretical. However we now give an explicit construction for the minimal method of choosing the weak hypothesis for which the number of iterations of greedy covering and InfoBoost grow linearly in $k$ and the number of iterations of AdaBoost with Bias is quadratic in $k$.

For any dimension $N$ we define an example set which is the rows of the following $(N+1) \times N$ dimensional matrix $\boldsymbol{x}$: All entries on the main diagonal and above are $+1$ and the remaining entries $-1$. In particular, the last row is all $-1$ (See Figure 4). The $i$-th instance $\boldsymbol{x}_i$ is the $i$-th row of this matrix and the first $N$ examples (rows) are labeled $+1$ and the label of the last row $y_{N+1}$ is $-1$.

Clearly the literal $X_N$ is consistent with the labels and thus always has error $0$ w.r.t. any distribution on the examples. But note that the disjunction of the last $k$

literals is also consistent (for any $k$). We will construct a distribution on the rows that gives high probability to the early rows (See Figure 4) and allows the following "minimal" choice of weak hypotheses: At iteration $t$, AdaBoost with Bias is given the weak hypothesis $X_t$. This weak hypothesis will have error $\frac{1}{2} - \frac{1}{2k}$ ($\delta = \frac{1}{2k}$) w.r.t. current distribution of the examples.

Contrary to our construction, the initial distribution for boosting applications is typically uniform. However, using padding this can be avoided but makes the construction more complicated. For any precision parameter $\epsilon \in (0, 1)$ and disjunction size $k$, we define the dimension

$$N := \left\lceil \frac{-\ln \frac{1}{2\varepsilon} - \ln\left(1 - \frac{1}{k}\right)}{\ln\left(1 - \frac{2}{k(k+1)}\right)} \right\rceil + 1 \geq \frac{k^2}{2}\ln\frac{1}{2\varepsilon} - \frac{k}{2}$$

The initial distribution $D_1$ is defined as

$$D_1(\boldsymbol{x}_t) := \begin{cases} \frac{1}{2k}, & \text{for } t = 1 \\ \frac{1}{k(k+1)}\left(1 - \frac{1}{k}\right)\left(1 - \frac{2}{k(k+1)}\right)^{t-2}, & \text{for } 2 \leq t \leq N - 1 \\ \frac{1}{2} - \sum_{t=1}^{N-1} D(\boldsymbol{x}_t), & \text{for } t = N \\ \frac{1}{2}, & \text{for t=}N + 1. \end{cases}$$

The example $\boldsymbol{x}_N$ has the lowest probability w.r.t. $D_1$ (See Figure 4). However one can show that its probability is at least $\varepsilon$.

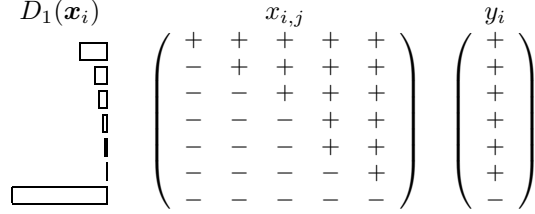

Figure 4: The examples (rows of the matrix), the labels, and the distribution $D_1$.

Also for $t \leq N - 1$, the probability $D_1(\boldsymbol{x}_{\leq t})$ of the first $t$ examples is $\frac{1}{2}\left\{1 - \left(1 - \frac{1}{k}\right)\left(1 - \frac{2}{k(k+1)}\right)^{t-1}\right\}$. AdaBoost with Bias does two update steps at iteration $t$ (constrain the error of $h_t$ to half and then sequentially the error of $\mathbf{1}$ to half.)

$$\widetilde{D}_t(i) = \frac{D_t(i)\exp\{-y_i h_t(\boldsymbol{x}_i)\alpha_t\}}{Z_t} \text{ and } D_{t+1} = \frac{\widetilde{D}_t(i)\exp\{-y_i\widetilde{\alpha}_t\}}{\widetilde{Z}_t}.$$

The $Z$'s are normalization factors, $\alpha_t = \frac{1}{2}\ln\frac{1-\varepsilon_t}{\varepsilon_t}$ and $\widetilde{\alpha}_t = \frac{1}{2}\ln\frac{\widetilde{D}_t(\boldsymbol{x}_{\leq N})}{\widetilde{D}_t(\boldsymbol{x}_{N+1})}$. The final hypothesis is the sign of the following linear combination: $H(\boldsymbol{x}) = \sum_{t=1}^{T}\alpha_t h_t(\boldsymbol{x}) + \sum_{t=1}^{T}\widetilde{\alpha}_t$.

**Proposition 1.** *For AdaBoost with Bias and $t \leq N$, $\Pr_{D_t}[X_t(\boldsymbol{x}_i) \neq y_i] = \frac{1}{2} - \frac{1}{2k}$.*

*Proof.* (Outline) Since each literal $X_t$ is one-sided, $X_t$ classifies the negative example $x_{N+1}$ correctly. Since $\Pr_{D_t}[X_t(\boldsymbol{x}_i) = y_i] = D_t(\boldsymbol{x}_{N+1}) + D_t(\boldsymbol{x}_{\leq t})$ and $D_t(\boldsymbol{x}_{N+1}) = \frac{1}{2}$

it suffices to show that $D_t(\boldsymbol{x}_{\leq t}) = \frac{1}{2k}$ for $t \leq N$. The proof is by induction on $t$. For $t = 1$, the statement follows from the definition of $D_1$. Now assume that the statement holds for any $t' < t$. Then we have

$$D_t(\boldsymbol{x}_{\leq t}) = D_t(\boldsymbol{x}_{\leq t-1}) + D_t(\boldsymbol{x}_t) = D_{t-1}(\boldsymbol{x}_{\leq t-1})\frac{e^{-\alpha_{t-1}}}{Z_{t-1}}\frac{e^{\widetilde{\alpha}_{t-1}}}{\widetilde{Z}_{t-1}} + D_t(\boldsymbol{x}_t). \quad (1)$$

Note that the example $\boldsymbol{x}_t$ is not covered by any previous hypotheses $X_1, \ldots, X_{t-1}$, and thus we have

$$D_t(\boldsymbol{x}_t) = D_1(\boldsymbol{x}_t) \prod_{j=1}^{t-1} \frac{e^{\alpha_j}}{Z_j}\frac{e^{\widetilde{\alpha}_j}}{\widetilde{Z}_j}. \quad (2)$$

Using the inductive assumption that $\Pr_{D_{t'}}[X_{t'}(\boldsymbol{x}_i) \neq y_i] = \frac{1}{2} - \frac{1}{2k}$, for $t' < t$, one can show that $\alpha_{t'} = \frac{1}{2}\ln\frac{k+1}{k-1}$, $Z_{t'} = \frac{1}{k}\sqrt{(k-1)(k+1)}$, $\widetilde{D}_{t'}(\boldsymbol{x}_{\leq N}) = \frac{1}{2} + \frac{1}{2(k+1)}$, $\widetilde{D}_{t'}(\boldsymbol{x}_{N+1}) = \frac{1}{2} - \frac{1}{2(k+1)}$, $\widetilde{\alpha}_{t'} = \frac{1}{2}\ln\frac{k+2}{k}$, and $\widetilde{Z}_{t'} = \frac{1}{k+1}\sqrt{k(k+2)}$. Substituting these values into the formulae (1) and (2), completes the proof. $\square$

**Theorem 2.** *For the described examples set, initial distribution $D_1$, and minimal choice of weak hypotheses, AdaBoost with Bias needs at least $N$ iterations to construct a final hypothesis whose error with respect to $D_1$ is below $\varepsilon$.*

*Proof.* Let $t$ be any integer smaller than $N$. At the end of the iteration $t$, the examples $\boldsymbol{x}_{t+1}, \ldots, \boldsymbol{x}_N$ are not correctly classified by the past weak hypotheses $X_1, \ldots, X_t$. In particular, the final linear combination evaluated at $\boldsymbol{x}_N$ is

$$H(\boldsymbol{x}_N) = \sum_{j=1}^{t}\alpha_j X_j(\boldsymbol{x}_N) + \sum_{j=1}^{t}\widetilde{\alpha}_j = -\sum_{j=1}^{t}\alpha_j + \sum_{j=1}^{t}\widetilde{\alpha}_j = -\frac{t}{2}\ln\frac{k+1}{k-1} + \frac{t}{2}\ln\frac{k+2}{k} < 0.$$

Thus $\text{sign}(H(\boldsymbol{x}_N)) = -1$ and the final hypothesis has error at least $D_1(\boldsymbol{x}_N) \geq \varepsilon$ with respect to $D_1$. $\square$

To show a similar lower bound for plain AdaBoost we use the same example set and the following sequence of weak hypotheses $X_1, \mathbf{1}, X_2, \mathbf{1}, \ldots X_N, \mathbf{1}$. For odd iteration numbers $t$ the above proposition shows the error of the weak hypothesis is $\frac{1}{2} - \frac{1}{2k}$ and for even iteration numbers one can show that the hypothesis $\mathbf{1}$ has error $\frac{1}{2} - \frac{1}{2(k+1)}$.

## 4 InfoBoost and SemiBoost for one-sided weak hypotheses

Aslam proved the following upper bound on the training error[Asl00] of InfoBoost:

**Theorem 3.** *The training error of the final hypothesis produced by InfoBoost is bounded by $\prod_{t=1}^{T}Z_t$, where $Z_t = \Pr_{D_t}[h_t(x_i) = +1]\sqrt{1 - \gamma_t[+1]^2} + \Pr_{D_t}[h_t(x_i) = -1]\sqrt{1 - \gamma_t[-1]^2}$ and edge[4] $\gamma_t[\pm 1] = 1 - 2\varepsilon_t[\pm 1]$.*

Let $\gamma_t = 1 - 2\varepsilon_t$. If $\gamma_t[+1] = \gamma_t[-1] = \gamma_t$, then $Z_t = \sqrt{1 - \gamma_t^2}$, as for AdaBoost. However, if $h_t$ is one-sided, InfoBoost gives the improved factor of $\sqrt{1 - \gamma_t}$:

**Corollary 4.** *For $t \geq 2$, if $h_t$ is one-sided w.r.t. $D_t$, then $Z_t = \sqrt{1 - \gamma_t}$.*

*Proof.* Wlog. assume $h_t$ is always correct when it predicts $+1$. Then $\gamma_t[+1] = 1$ and the first summand in the expression for $Z_t$ given in the above theorem disappears. Recall that InfoBoost maintains the distribution $D_t$ over examples so that $\Pr_{D_t}[y_i = +1] = \frac{1}{2}$ for $t \geq 2$. So the second summand becomes

$$2\sqrt{\Pr_{D_t}[h_t(x_i) = -1, y_i = +1]\Pr_{D_t}[h_t(x_i) = -1, y_i = -1]}$$

$$= 2\sqrt{\Pr_{D_t}[y_i = +1]\Pr_{D_t}[y_i = -1]}\sqrt{\Pr_{D_t}[h_t(x_i) = -1|y_i = +1]\Pr_{D_t}[h_t(x_i) = -1|y_i = -1]}$$

$$= \sqrt{\Pr_{D_t}[h_t(x_i) = -1|y_i = +1]}.$$

By the definition of $\gamma_t$, we have

$$\begin{aligned}
1 - \gamma_t &= 2\Pr_{D_t}[h_t(x_i) \neq y_i] \\
&= 2\Pr_{D_t}[h_t(x_i) = -1, y_i = +1] \quad \text{(because of one-sidedness of } h_t) \\
&= 2\Pr_{D_t}[y_i = +1]\Pr_{D_t}[h_t(x_i) = -1|y_i = +1] \\
&= \Pr_{D_t}[h_t(x_i) = -1|y_i = +1] \quad \text{(because } \Pr_{D_t}[y_i = +1] = \frac{1}{2})
\end{aligned}$$

$\square$

This corollary implies that if a one-sided hypothesis is chosen at each iteration, then InfoBoost constructs a final hypothesis consistent with all $m$ examples within $\frac{2}{\gamma}\ln m$ iterations. When the considered weak hypotheses are positively one-sided, then the trivial greedy covering algorithm (which simply chooses the set that covers the most uncovered positive examples), achieves the improved factor of $1 - \gamma$, which means at most $\frac{1}{\gamma}\ln m$ iterations. By a careful analysis (not included), one can show that the factor for InfoBoost can be improved to $1 - \gamma$, if all weak hypotheses are one-sided. So in this case InfoBoost indeed matches the $1 - \gamma$ factor of the greedy covering algorithm.

A technical problem arises when InfoBoost is given a set of examples that are all labeled $+1$. Then we have $\alpha_1[+1] = \infty$ and $\alpha_1[-1] = -\infty$. This implies $H(x) = \alpha_1[h_1(x_i)]h_t(x_i) = \infty$ for any instance $x_i$. Thus InfoBoost terminates in a single iteration and outputs a hypothesis that predicts $+1$ for any instance and InfoBoost cannot be used for constructing a cover.

We propose a natural way to cope with this subtlety. Recall that the final hypothesis of InfoBoost is given by $H(x) = \sum_{t=1}^{T} \alpha_t[h_t(x)] \; h_t(x)$. This doesn't seem to be a linear combination of hypotheses from $\mathcal{H}$ since the coefficients vary with the prediction of weak hypotheses. However observe that $\alpha_t[h_t(x)] \; h_t(x) = \alpha_t[+1] \; h_t^+(x) + \alpha_t[-1] \; h_t^-(x)$, where $h^{\pm} = h(x)$ if $h(x) = \pm 1$ and $0$ otherwise. We call $h^+$ and $h^-$ the *semi* hypotheses of $h$. Note that $h^+(x) = \frac{h(x)+1}{2}$ and $h^-(x) = \frac{h(x)-1}{2}$. So the final hypothesis of InfoBoost and the new algorithm we will define in a moment is a bias plus a linear combination of the the original weak learners in $\mathcal{H}$.

We propose the following variant of AdaBoost (called *Semi-Boost*): In each iteration execute one step of AdaBoost but the chosen weak hypothesis must be a semi hypothesis of one of the original hypothesis $h \in \mathcal{H}$ which has a positive edge. SemiBoost avoids the outlined technical problem and can handle equally labeled example sets. Also if all the chosen hypotheses are of the $h^+$ type then the final hypothesis is a disjunction. If hypotheses are chosen by smallest error (largest edge), then the greedy covering algorithm is simulated. Analogously, if all the chosen hypotheses are of the $h^-$ type then one can show that the final hypothesis of SemiBoost is a conjunction. Furthermore, two steps of SemiBoost (with hypothesis $h^+$ in the first step followed by the sibling hypothesis $h^-$ in the second step) are equivalent to one step of InfoBoost with hypothesis $h$.

Finally we note that the final hypothesis of InfoBoost (or SemiBoost) is not well-defined when it includes both types of one-sided hypotheses, i.e. positive and negative infinite coefficients may conflict each other. We propose two solutions. First, following [SS99] one can use the modified coefficients $\alpha[\pm 1]' = \frac{1}{2} \ln \frac{1-\varepsilon[\pm 1]+\Delta}{\varepsilon[\pm 1]+\Delta}$ for small $\Delta > 0$. It can be shown that the new $Z'$ increases by at most $\sqrt{2\Delta}$([SS99]). Second, we allow infinite coefficients but interpret the final hypothesis as a version of a decision list [Riv87]: Whenever more than one semi hypotheses with infinite coefficients are non-zero on the current instance, then the semi hypothesis with the lowest iteration number determines the label. Once such a consistent decision list over some set of hypothesis $h_t$ and $\mathbf{1}$ has been found, it is easy the find an alternate linear combination of the same set of hypotheses (using linear programming) that maximizes the margin or minimizes the one-norm of the coefficient vector subject to consistency.

**Conclusion:** We showed that AdaBoost can require significantly more iterations than the simple greedy cover algorithm when the weak hypotheses are one-sided and gave a variant of AdaBoost that can readily exploit one-sidedness. The open question is whether the new SemiBoost algorithm gives improved performance on natural data and can be used for feature selection.

**Acknowledgment:** This research benefited from many discussions with Gunnar Rätsch. He encouraged us to analyze AdaBoost with Bias and suggested to write the final hypothesis of InfoBoost as a linear combination of semi hypotheses. We also thank anonymous referees for helpful comments.

## Footnotes

[1]This assumes equal weight on both types of examples.

[2]Wipe out the weights of positive examples that are correctly classified and re-balance both types of examples.

[3]For the sake of simplicity we focus on the case when the range of the labels and the weak hypotheses is $\pm 1$ valued. Many parts of this paper generalize to the range $[-1, 1]$ [SS99, Asl00].

[4]The edge $\gamma$ and error $\epsilon$ are related as follows: $\gamma = 1 - 2\epsilon$ and $\epsilon = \frac{1}{2} - \frac{1}{2\gamma}$; $\epsilon = \frac{1}{2} \Leftrightarrow \gamma = 0$.

# References

[Asl00] J. A. Aslam. Improving algorithms for boosting. In *Proc. 13th Annu. Conference on Comput. Learning Theory*, pages 200–207, 2000.

[Fre95] Y. Freund. Boosting a weak learning algorithm by majority. *Inform. Comput.*, 121(2):256–285, September 1995. Also appeared in COLT90.

[FS97] Y. Freund and R. E. Schapire:. A decision-theoretic generalization of on-line learning and an application to boosting. *J. Comput. Syst. Sci.*, 55(1):119–139, 1997.

[KW99] Jyrki Kivinen and Manfred K. Warmuth. Boosting as entropy projection. In *Proc. 12th Annu. Conf. on Comput. Learning Theory*, pages 134–144. ACM Press, New York, NY, 1999.

[Laf99] J. Lafferty. Additive models, boosting, and inference for generalized divergences. In *Proc. 12th Annu. Conf. on Comput. Learning Theory*, pages 125–133. ACM, 1999.

[MG92] A. A. Razborov M. Goldmann, J. Hastad. Majority gates vs. general weighted threshold gates. *Journal of Computation Complexity*, 1(4):277–300, 1992.

[Nat91] B. K. Natarajan. *Machine Learning: A Theoretical Approach*. Morgan Kaufmann, San Mateo, CA, 1991.

[Riv87] R. L. Rivest. Learning decision lists. *Machine Learning*, 2:229–246, 1987.

[RW02] G. Rätsch and M. K. Warmuth. Maximizing the margin with boosting. In *Proceedings of the 15th Annual Conference on Computational Learning Theory*, pages 334–350. Springer, July 2002.

[SS99] Robert E. Schapire and Yoram Singer. Improved boosting algorithms using confidence-rated predictions. *Machine Learning*, 37(3):297–336, 1999.
